# Multiple Instance Boosting for Object Detection

**Paul Viola, John C. Platt, and Cha Zhang**
Microsoft Research
1 Microsoft Way
Redmond, WA 98052
{viola,jplatt}@microsoft.com

## Abstract

A good image object detection algorithm is accurate, fast, and does not require exact locations of objects in a training set. We can create such an object detector by taking the architecture of the Viola-Jones detector cascade and training it with a new variant of boosting that we call MIL-Boost. MILBoost uses cost functions from the Multiple Instance Learning literature combined with the AnyBoost framework. We adapt the feature selection criterion of MILBoost to optimize the performance of the Viola-Jones cascade. Experiments show that the detection rate is up to 1.6 times better using MILBoost. This increased detection rate shows the advantage of simultaneously learning the locations and scales of the objects in the training set along with the parameters of the classifier.

## 1   Introduction

When researchers use machine learning for object detection, they need to know the location and size of the objects, in order to generate positive examples for the classification algorithm. It is often extremely tedious to generate large training sets of objects, because it is not easy to specify exactly where the objects are. For example, given a ZIP code of handwritten digits, which pixel is the location of a "5" ? This sort of ambiguity leads to training sets which themselves have high error rates, this limits the accuracy of any trained classifier.

In this paper, we explicitly acknowledge that object recognition is innately a Multiple Instance Learning problem: we know that objects are located in regions of the image, but we don't know exactly where. In MIL, training examples are not singletons. Instead, they come in "bags", where all of the examples in a bag share a label [4]. A positive bag means that at least one example in the bag is positive, while a negative bag means that all examples in the bag are negative. In MIL, learning must simultaneously learn which examples in the positive bags are positive, along with the parameters of the classifier.

We have combined MIL with the Viola-Jones method of object detection, which uses Adaboost [11] to create a cascade of detectors. To do this, we created MILBoost, a new method for folding MIL into the AnyBoost [9] framework. In addition, we show how early stage in the detection cascade can be re-trained using information extracted from the final MIL classifier.

We test this new form of MILBoost for detecting people in a teleconferencing application.

This is a much harder problem then face detection, since the participants do not look at the camera (and sometimes away). The MIL framework is shown to produce classifiers with much higher detection rates and fast computation times.

## 1.1 Structure of paper

We first review the previous work in two fields: previous related work in object detection (Section 2.1) and in multiple instance learning (Section 2.2). We derive a new MIL variant of boosting in Section 3, called MILBoost. MILBoost is used to train a detector in the Viola-Jones framework in Section 4. We then adapt MILBoost to train an effective cascade using a new criterion for selecting features in the early rounds of training (Section 5). The paper concludes in Section 6 with experimental results on the problem of person detection in a teleconferencing application. The MIL framework is shown to produce classifiers with much higher detection rates and fast computation times.

## 2 Relationship to previous work

This paper lies at the intersection between the subfields of object detection and multiple instance learning. Therefore, we discuss the relationship with previous work in each subfield separately.

### 2.1 Previous work in image object detection

The task of object detection in images is quite daunting. Amongst the challenges are 1) creating a system with high accuracy and low false detection rate, 2) restricting the system to consume a reasonable amount of CPU time, and 3) creating a large training set that has low labeling error.

Perona et. al [3, 5] and Schmid [12] have proposed constellation models: spatial models of local image features. These models can be trained using unsegmented images in which the object can appear at any location. Learning uses EM-like algorithms to iteratively localize and refine discriminative image features. However, hitherto, the detection accuracy has not be as good as the best methods.

Viola and Jones [13] created a system that exhaustively scans pose space for generic objects. This system is accurate, because it is trained using AdaBoost [11]. It is also very efficient, because it uses a cascade of detectors and very simple image features. However, the AdaBoost algorithm requires exact positions of objects to learn.

The closest work to this paper is Nowlan and Platt [10], which built on the work of Keeler, et. al [7] (see below). In the Nowlan paper, a convolutional neural network was trained to detect hands. The exact location and size of the hands is approximately truthed: the neural network used MIL training to co-learn the object location and the parameters of the classifier. The system is effective, but is not as fast as Viola and Jones, because the detector is more complex and it does not use a cascade.

This paper builds on the accuracy and speed of Viola and Jones, by using the same architecture. We attempt to gain the flexibility of the constellation models. Instead of an EM-like algorithm, we use MIL to create our system, which does not require iteration. Unlike Nowlan and Platt, we maintain a cascade of detectors for maximum speed.

### 2.2 Previous work in Multiple Instance Learning

The idea for multiple instance learning was originally proposed 1990 for handwritten digit recognition by Keeler, et. al [7]. Keeler's approach was called Integrated Segmentation

and Recognition (ISR). In that paper, the position of a digit in a ZIP code was considered completely unknown. ISR simultaneously learned the positions of the digits and the parameters of a convolutional neural network recognizer. More details on ISR are given below (Section 3.2).

Another relevant example of MIL is the Diverse Density approach of Maron [8]. Diverse Density uses the Noisy OR generative model [6] to explain the bag labels. A gradient-descent algorithm is used to find the best point in input space that explains the positive bags. We also utilize the Noisy OR generative model in a version of our algorithm, below (Section 3.1).

Finally, a number of researchers have modified the boosting algorithm to perform MIL. For example, Andrews and Hofmann [1] have proposed modifying the inner loop of boosting to use linear programming. This is not practically applicable to the object detection task, which can have millions of examples (pixels) and thousands of bags.

Another approach is due to Auer and Ortner [2], which enforces a constraint that weak classifiers must be either hyper-balls or hyper-rectangles in $\Re^N$. This would exclude the fast features used by Viola and Jones.

A third approach is that of Xu and Frank [14], which uses a generative model that the probability of a bag being positive is the mean of the probabilities that the examples are positive. We believe that this rule is unsuited for object detection, because only a small subset of the examples in the bag are ever positive.

## 3 MIL and Boosting

We will present two new variants of AdaBoost which attempt to solve the MIL problem. The derivation uses the AnyBoost framework of of Mason et al. which views boosting as a gradient descent process [9]. The derivation builds on previous appropriate MIL cost functions, namely ISR and Noisy OR. The Noisy OR derivation is simpler and a bit more intuitive.

### 3.1 Noisy-OR Boost

Recall in boosting each example is classified by a linear combination of weak classifiers. In MILBoost, examples are not individually labeled. Instead, they reside in bags. Thus, an example is indexed with two indices: $i$, which indexes the bag, and $j$, which indexes the example within the bag. The score of the example is $y_{ij} = C(x_{ij})$ and $C(x_{ij}) = \sum_t \lambda_t c^t(x_{ij})$ a weighted sum of weak classifiers. The probability of an example is positive is given by

$$p_{ij} = \frac{1}{1 + \exp(-y_{ij})},$$

the standard logistic function. The probability that the bag is positive is a "noisy OR" $p_i = 1 - \prod_{j \in i}(1 - p_{ij})$ [6] [8]. Under this model the likelihood assigned to a set of training bags is:

$$L(C) = \prod_i p_i^{t_i}(1 - p_i)^{(1-t_i)}$$

where $t_i \in \{0, 1\}$ is the label of bag $i$.

Following the AnyBoost approach, the weight on each example is given as the derivative of the cost function with respect to a change in the score of the example. The derivative of the log likelihood is:

$$\frac{\partial \log L(C)}{\partial y_{ij}} = w_{ij} = \frac{t_i - p_i}{p_i} p_{ij}. \tag{1}$$

Note, that the weights here are signed. The interpretation is straightforward; the sign determines the example label. Each round of boosting is a search for a classifier which maximizes $\sum_{ij} c(x_{ij})w_{ij}$ where $c(x_{ij})$ is the score assigned to the example by the weak classifier (for a binary classifier $c(x_{ij}) \in \{-1, +1\}$). The parameter $\lambda_t$ is determined using a line search to maximize $\log L(C + \lambda_t c_t)$.

Examining the criteria (1) the weight on each example is the product of two quantities: the bag weight $W_{\text{bag}} = \frac{t_i - p_i}{p_i}$ and the instance weight $W_{\text{instance}} = p_{ij}$. Observe that $W_{\text{bag}}$ for a negative bag is always $-1$. Thus, the weight for a negative instance, $p_{ij}$, is the same that would result in a non-MIL AdaBoost framework (i.e. the negative examples are all equally negative). The weight on the positive instances is more complex. As learning proceeds and the probability of the bag approaches the target, the weight on the entire bag is reduced. Within the bag, the examples are assigned a weight which is higher for examples with higher scores. Intuitively the algorithm selects a subset of examples to assign a higher positive weight, and these example dominate subsequent learning.

### 3.2 ISR Boost

The authors of the ISR paper may well have been aware of the Noisy OR criteria described above. They chose instead to derive a different perhaps less probabilistic criteria. They do this in part because the derivatives (and hence example weights) lead to a form of instance competition.

Define $\chi_{ij} = \exp(y_{ij})$, $S_i = \sum_{j \in i} \chi_{ij}$ and $p_i = \frac{S_i}{1+S_i}$. Keeler et al. argue that $\chi_{ij}$ can be interpreted as the likelihood that the object occurs at $ij$. The quantity $S_i$ can be interpreted as a likelihood ratio that *some* (at least one) instance is positive, and finally $p_i$ is the probability that some instance is positive. The example weights for the ISR framework are:

$$\frac{\partial \log L(C)}{\partial y_{ij}} = w_{ij} = (t_i - p_i)\frac{\chi_{ij}}{\sum_{j \in i} \chi_{ij}} \tag{2}$$

Examining the ISR criteria reveals two key properties. The first is the form of the example weight which is explicitly competitive. The examples in the bag compete for weight, since the weight is normalized by sum of the $\chi_{ij}$'s. Though the experimental evidence is weak, this rule perhaps leads to a very localized representation, where a single example is labeled positive and the other examples are labeled negative. The second property is that the negative examples also compete for weight. This turns out to be troublesome in the detection framework since there are many, many more negative examples than positive. How many negative bags should there be? In contrast, the Noisy OR criteria treats all negative examples as independent negative examples.

## 4  Application of MIL Boost to Object Detection in Images

Each image is divided into a set of overlapping square windows that uniformly sample the space of position and scale (typically there are between 10,000 and 100,000 windows in a training image). Each window is used as an example for the purposes of training and detection. Each training image is labeled to determine the position and scale of the object of interest. For certain types of objects, such as frontal faces, it may be possible to accurately determine the position and scale of the face. One possibility is to localize the eyes and then to determine the single positive image window in which the eyes appear at a given relative location and scale. Even for this type of object the effort in carefully labeling the images is significant.

For many other types of objects (objects which may be visible from multiple poses, or

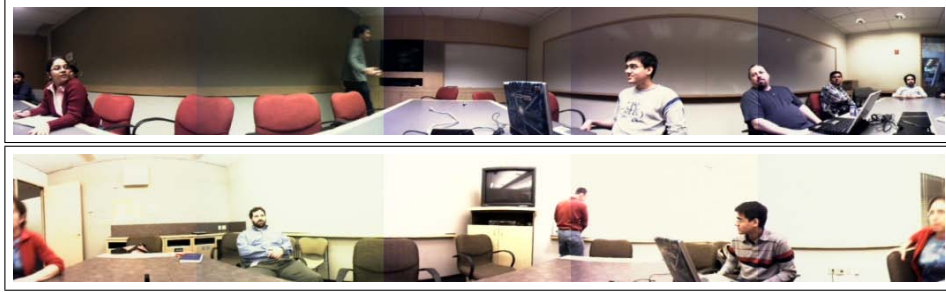

Figure 1: Two example images with people in a wide variety of poses. The algorithm will attempt to detect *all* people in the images, including those that are looking away from the camera.

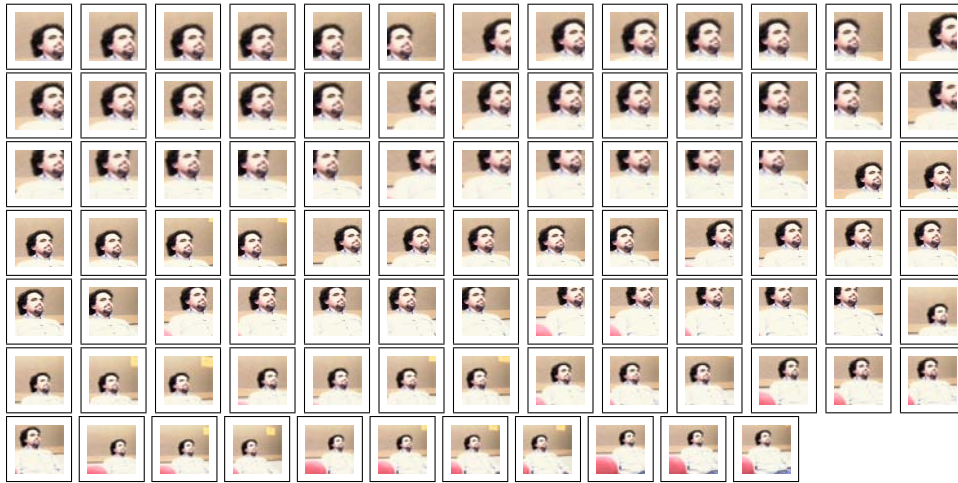

Figure 2: Some of the subwindows in one positive bag.

are highly varied, or are flexible) the "correct" geometric normalization is unclear. It is not clear how to normalize images of people in a conference room, who may be standing, sitting upright, reclining, looking toward, or looking away from the camera. Similar questions arise for most other image classes such as cars, trees, or fruit.

Experiments in this paper are performed on a set of images from a teleconferencing application. The images are acquired from a set of cameras near the center of the conference room (see Figure 1). The practical challenge is to steer a synthetic virtual camera toward the location of the speaker. The focus here is on person detection; determination of the person who is speaking is beyond the scope of this paper.

In every training image each person is labeled by hand. The labeler is instructed to draw a box around the head of the person. While this may seem like a reasonable geometric normalization, it ignores one critical issue, *context*. At the available resolution (approximately 1000x150 pixels) the head is often less than 10 pixels wide. At this resolution, even for clear frontal faces, the best face detection algorithms frequently fail. There are simply too few pixels on the face. The only way to detect the head is to include the surrounding image context. It is difficult to determine the correct quantity of image context (Figure 2 shows many possible normalizations).

If the body context is used to assist in detection, it is difficult to foresee the effect of body pose. Some of the participants are facing right, others left, and still others are leaning far forward/backward (while taking notes or reclining). The same context image is not be appropriate for all situations.

Both of these issues can be addressed with the use of MIL. Each positive head is represented, during training, by a large number of related image windows (see Figure 2). The MIL boosting algorithm is then used to *simultaneously* learn a detector *and* determine the location and scale of the appropriate image context.

## 5 MIL Boosting a Detection Cascade

In their work on face detection Viola and Jones train a cascade of classifiers, each designed to achieve high detection rates and modest false positive rates. During detection almost all of the computation is performed by the early stages in the cascade, perhaps 90% in the first 10 features. Training the initial stages of the cascade is the key to a fast and effective classifier.

Training and evaluating a detector in a MIL framework has a direct impact on cascade construction, both on the features selected and the appropriate thresholds.

The result of the MIL boost learning process is not only an example classifier, but also a set of weights on the examples. Those examples in positive bags which are assigned high weight have also high score. The final classifier labels these examples positive. The remaining examples in the positive bags are assigned a low weight and have a low score. The final classifier often classifies these examples as negative (as they should be).

Since boosting is a greedy process, the initial weak classifiers do not have any knowledge of the subsequent classifiers. As a result, the first classifiers selected have no knowledge of the final weights assigned to the examples. The key to efficient processing, is that the initial classifiers have a low false negative rate *on the examples determined to be positive by the final MIL classifier*.

This suggests a simple scheme for retraining the initial classifiers. Train a complete MIL boosted classifier and set the detection threshold to achieve the desired false positive and false negative rates. Retrain the initial weak classifier so that it has a zero false negative rate on the examples *labeled positive by the full classifier*. This results in a significant increase in the number of examples which can be pruned by this classifier. The process can be repeated, so that the second classifier is trained to yield a zero false negative rate on the *remaining* examples.

## 6 Experimental Results

Experiments were performed using a set of 8 videos recorded in different conference rooms. A collection of 1856 images were sampled from these videos. In all cases the detector was trained on 7 video conferences and tested on the remaining video conference. There were a total of 12364 visible people in these images. Each was labeled by drawing a rectangle around the head of each person.

Learning is performed on a total of about 30 million subwindows in the 1856 images. In addition to the monochrome images, two additional feature images are used. One measures the difference from the running mean image (this is something like background subtraction) and the other measures temporal variance over longer time scales. A set of 2654 rectangle filters are used for training. In each round the optimal filter and threshold is selected. In each experiment a total of 60 filters are learned.

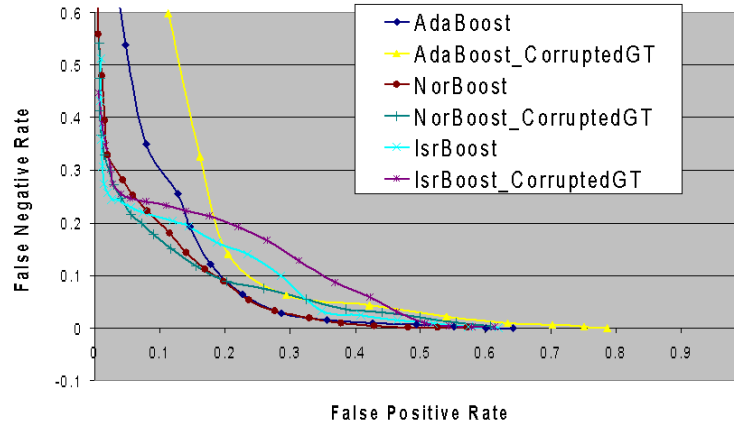

Figure 3: ROC comparison between various boosting rules.

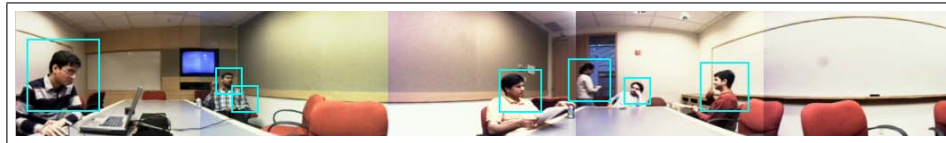

Figure 4: One example from the testing dataset and overlaid results.

We compared classical AdaBoost with two variants of MIL boost: ISR and Noisy-OR. For the MIL algorithms there is one bag for each labeled head, containing those positive windows which overlap that head. Additionally there is one negative bag for each image. After training, performance is evaluated on held out conference video (see Figure 3).

During training a set of positive windows are generated for each labeled example. All windows whose width is between 0.67 times and 1.5 times the head width *and* whose center is within 0.5 times the head width of the center of the head are labeled positive. An exception is made for AdaBoost, which has a tighter definition on positive examples (width between 0.83 and 1.2 times the head width and center within 0.2 times the head width) and produces better performance than the looser criterion. All windows which do not overlap with any head are considered negative. For each algorithm one experiment uses the ground truth obtained by hand (which has small yet unavoidable errors). A second experiment corrupts this ground truth further, moving each head by a uniform random shift such that there is non-zero overlap with the true position. Note that conventional AdaBoost is much worse when trained using corrupted ground truth. Interestingly, Adaboost is worse than NorBoost using the "correct" ground truth, even with a tight definition of positive examples. We conjecture that this is due to unavoidable ambiguity in the training and testing data.

Overall the MIL detection results are practically useful. A typical example of detection results are shown in Figure 4. Results shown are for the noisy OR algorithm. In order to simplify the display, significantly overlapping detection windows are averaged into a single window.

The scheme for retraining the initial classifier was evaluated on the noisy OR strong classifier trained above. Training a conventional cascade requires finding a small set of weak classifiers that can achieve zero false negative rate (or almost zero) and a low false positive rate. Using the first weak classifier yields a false positive rate of 39.7%. Including the first

four weak classifiers yields a false positive rate of 21.4%. After retraining the first weak classifier alone yields a false positive rate of 11.7%. This improved rejection rate has the effect of reducing computation time of the cascade by roughly a factor of three.

## 7    Conclusions

This paper combines the truthing flexibility of multiple instance learning with the high accuracy of the boosted object detector of Viola and Jones. This was done by introducing a new variant of boosting, called MILBoost. MILBoost combines examples into bags, using combination functions such as ISR or Noisy OR. Maximum likelihood on the output of these bag combination functions fit within the AnyBoost framework, which generates boosting weights for each example.

We apply MILBoost to Viola-Jones face detection, where the standard AdaBoost works very well. NorBoost improves the detection rate over standard AdaBoost (tight positive) by nearly 15% (at a 10% false positive rate). Using MILBoost for object detection allows the detector to flexibly assign labels to the training set, which reduces label noise and improves performance.

## References

[1] S. Andrews and T. Hofmann. Multiple-instance learning via disjunctive programming boosting. In S. Thrun, L. K. Saul, and B. Schölkopf, editors, *Proc. NIPS*, volume 16. MIT Press, 2004.

[2] P. Auer and R. Ortner. A boosting approach to multiple instance learning. In *Lecture Notes in Computer Science*, volume 3201, pages 63–74, October 2004.

[3] M. C. Burl, T. K. Leung, and P. Perona. Face localization via shape statistics. In *Proc. Int'l Workshop on Automatic Face and Gesture Recognition*, pages 154–159, 1995.

[4] T. G. Dietterich, R. H. Lathrop, and T. Lozano-Pérez. Solving the multiple instance problem with axis-parallel rectangles. *Artif. Intell.*, 89(1-2):31–71, 1997.

[5] R. Fergus, P. Perona, and A. Zisserman. Object class recognition by unsupervised scale-invariant learning. In *Proc. CVPR*, volume 2, pages 264–271, 2003.

[6] D. Heckerman. A tractable inference algorithm for diagnosing multiple diseases. In *Proc. UAI*, pages 163–171, 1989.

[7] J. D. Keeler, D. E. Rumelhart, and W.-K. Leow. Integrated segmentation and recognition of hand-printed numerals. In *NIPS-3: Proceedings of the 1990 conference on Advances in neural information processing systems 3*, pages 557–563, San Francisco, CA, USA, 1990. Morgan Kaufmann Publishers Inc.

[8] O. Maron and T. Lozano-Perez. A framework for multiple-instance learning. In *Proc. NIPS*, volume 10, pages 570–576, 1998.

[9] L. Mason, J. Baxter, P. Bartlett, and M. Frean. Boosting algorithms as gradient descent in function space, 1999.

[10] S. J. Nowlan and J. C. Platt. A convolutional neural network hand tracker. In G. Tesauro, D. Touretzky, and T. Leen, editors, *Advances in Neural Information Processing Systems*, volume 7, pages 901–908. The MIT Press, 1995.

[11] R. E. Schapire and Y. Singer. Improved boosting algorithms using confidence-rated predictions. In *Proc. COLT*, volume 11, pages 80–91, 1998.

[12] C. Schmid and R. Mohr. Local grayvalue invariants for image retrieval. *IEEE Trans. PAMI*, 19(5):530–535, 1997.

[13] P. Viola and M. Jones. Robust real-time object detection. *Int'l. J. Computer Vision*, 57(2):137–154, 2002.

[14] X. Xu and E. Frank. Logistic regression and boosting for labeled bags of instances. In *Lecture Notes in Computer Science*, volume 3056, pages 272–281, April 2004.
